# Sequential Hypothesis Testing under Stochastic Deadlines

**Peter I. Frazier**
ORFE
Princeton University
Princeton, NJ 08544
pfrazier@princeton.edu

**Angela J. Yu**
CSBMB
Princeton University
Princeton, NJ 08544
ajyu@princeton.edu

## Abstract

Most models of decision-making in neuroscience assume an *infinite* horizon, which yields an optimal solution that integrates evidence up to a fixed decision threshold; however, under most experimental as well as naturalistic behavioral settings, the decision has to be made before some finite deadline, which is often experienced as a stochastic quantity, either due to variable external constraints or internal timing uncertainty. In this work, we formulate this problem as sequential hypothesis testing under a stochastic horizon. We use dynamic programming tools to show that, for a large class of deadline distributions, the Bayes-optimal solution requires integrating evidence up to a threshold that declines monotonically over time. We use numerical simulations to illustrate the optimal policy in the special cases of a fixed deadline and one that is drawn from a gamma distribution.

## 1 Introduction

Major strides have been made in understanding the detailed dynamics of decision making in simple two-alternative forced choice (2AFC) tasks, at both the behavioral and neural levels. Using a combination of probabilistic and dynamic programming tools, it has been shown that when the decision horizon is infinite (*i.e.* no deadline), the optimal policy is to accumulate sensory evidence for one alternative versus the other until a fixed threshold, and report the corresponding hypothesis [1]. Under similar experimental conditions, it appears that humans and animals accumulate information and make perceptual decisions in a manner close to this optimal strategy [2–4], and that neurons in the posterior parietal cortex exhibit response dynamics similar to that prescribed by the optimal algorithm [6]. However, in most 2AFC experiments, as well as in more natural behavior, the decision has to be made before some finite deadline. This corresponds to a finite-horizon sequential decision problem. Moreover, there is variability associated with that deadline either due to external variability associated with the deadline imposition itself, or due to internal timing uncertainty about how much total time is allowed and how much time has already elapsed. In either case, with respect to the observer's *internal* timer, the deadline can be viewed as a stochastic quantity.

In this work, we analyze the optimal strategy and its dynamics for decision-making under the pressure of a stochastic deadline. We show through analytical and numerical analysis that the optimal policy is a monotonically declining decision threshold over time. A similar result for deterministic deadlines was shown in [5]. Declining decision thresholds have been used in [7] to model the speed vs. accuracy tradeoff, and also in the context of sequential hypothesis testing ( [8]). We first present a formal model of the problem, as well as the main theoretical results (Sec. 2). We then use numerical simulations to examine the optimal policy in some specific examples (Sec. 3).

## 2 Decision-making under a Stochastic Deadline

We assume that on each trial, a sequence of i.i.d inputs are observed: $x^1, x^2, x^3, \ldots$. With probability $p^0$, all the inputs for the trial are generated from a probability density $f_1$, and, with probability

$1 - p^0$, they are generated from an alternate probability density $f_0$. Let $\theta$ be index of the generating distribution. The objective is to decide whether $\theta$ is 0 or 1 quickly and accurately, while also under the pressure of a stochastic decision deadline.

We define $\mathbf{x}^t \triangleq (x^1, x^2, \ldots, x^t)$ to be the vector of observations made by time $t$. This vector of observations gives information about the generating density $\theta$. Defining $p^t \triangleq \mathbb{P}\{\theta = 1 \mid \mathbf{x}^t\}$, we observe that $p^{t+1}$ may be obtained iteratively from $p^t$ via Bayes' rule,

$$p^{t+1} = \mathbb{P}\{\theta = 1 \mid \mathbf{x}^{t+1}\} = \frac{p^t f_1(x^{t+1})}{p^t f_1(x^{t+1}) + (1 - p^t) f_0(x^{t+1})}. \tag{1}$$

Let $D$ be a deadline drawn from a *known* distribution that is independent of the observations $\mathbf{x}^t$. We will assume that the deadline $D$ is observed immediately and effectively terminates the trial. Let $c > 0$ be the cost associated with each unit time of decision delay, and $d \geq .5$ be the cost associated with exceeding the deadline, where both $c$ and $d$ are normalized against the (unit) cost of making an incorrect decision. We choose $d \geq .5$ so that $d$ is never smaller than the expected penalty for guessing at $\theta$. This avoids situations in which we prefer to exceed the deadline.

A decision-policy $\pi$ is a sequence of mappings, one for each time $t$, from the observations so far to the set of possible actions: stop and choose $\theta = 0$; stop and choose $\theta = 1$; or continue sampling. We define $\tau_\pi$ to be the time when the decision is made to stop sampling under decision-policy $\pi$, and $\delta_\pi$ to be the hypothesis chosen at this time – both are random variables dependent on the sequence of observations. More formally, $\pi \triangleq \pi^0, \pi^1, \ldots$, where $\pi^t(\mathbf{x}^t) \mapsto \{0, 1, \text{continue}\}$, and $\tau_\pi \triangleq \min(D, \inf\{t \in \mathbb{N} : \pi^t(\mathbf{x}^t) \in \{0, 1\}\})$, $\delta_\pi \triangleq \pi^{\tau_\pi}(\mathbf{x}^{\tau_\pi})$. We may also define $\sigma_\pi \triangleq \inf\{t \in \mathbb{N} : \pi^t(\mathbf{x}^t) \in \{0, 1\}\}$ to be the time when the policy would choose to stop sampling if the deadline were to fail to occur. Then $\tau_\pi = \min(D, \sigma_\pi)$.

Our loss function is defined to be $l(\tau, \delta; \theta, D) = \mathbf{1}_{\{\delta \neq \theta\}} \mathbf{1}_{\{\tau < D\}} + c\tau + d\mathbf{1}_{\{\tau \geq D\}}$. The goal is to find a decision-policy $\pi$ which minimizes the total expected loss

$$L_\pi \triangleq \langle l(\tau_\pi, \delta_\pi; \theta, D) \rangle_{\theta, D, \mathbf{x}} = \mathbb{P}(\delta_\pi \neq \theta, \tau_\pi < D) + c\langle \tau_\pi \rangle + d\,\mathbb{P}(D \leq \tau_\pi). \tag{2}$$

## 2.1 Dynamic Programming

A decision policy is characterized by how $\tau$ and $\delta$ are generated as a function of the data observed so far. Thus, finding the optimal decision-policy is equivalent to finding the random variables $\tau$ and $\delta$ that minimize $\langle l(\tau, \delta; \theta, D) \rangle$. The optimal policy decides whether or not to stop based on whether $p^t$ is inside a set $C^t \subseteq [0, 1]$ or not. Our goal is to show that $C^t$ is a continuous interval, that $C^{t+1} \subseteq C^t$, and that for large enough $t$, $C^t$ is empty. That is, the optimal policy is to iteratively compute $p^t$ based on incoming data, and to decide for the respective hypothesis as soon as it hits either a high ($\delta = 1$) or low ($\delta = 0$) threshold. Furthermore, the two thresholds decay toward each other over time and eventually meet.

We will use tools from dynamic programming to analyze this problem. Our approach is illustrated in Fig. 2.1. The red line denotes the cost of stopping at time $t$ as a function of the current belief $p^t = p$. The blue line denotes the cost of continuing at least one more time step, as a function of $p^t$. The black line denotes the cost of continuing at least two more time steps, as a function of $p^t$. Because the cost of continuing is concave in $p^t$ (Lemma 1), and larger than stopping for $p^t \in \{0, 1\}$ (Lemma 4), the continuation region is an interval delimited by where the costs of continuing and stopping intersect (blue dashed lines). Moreover, because the cost of continuing two more timesteps is always larger than that of continuing one more for a given amount of belief (Lemmas 2 and 3), that "window" of continuation narrows over time (Main Theorem). This method of proof parallels that of optimality for the classic sequential probability ratio test in [10].

Before proving the lemmas and the theorem, we first introduce some additional definitions. The *value function* $V : \mathbb{N} \times [0, 1] \mapsto \mathbb{R}_+$ specifies the minimal cost (incurred by the optimal policy) at time $t$, given that the deadline has not yet occurred, that $\mathbf{x}^t$ have been observed, and that the current cumulative evidence for $\theta = 1$ is $p^t$: $V(t, p^t) \triangleq \inf_{\tau \geq t, \delta} \langle l(\tau, \delta; \theta, D) \mid D > t, p^t \rangle_{\theta, D, \mathbf{x}}$. The cost associated with continuing at time $t$, known as the *Q-factor for continuing* and denoted by $Q$, takes the form

$$Q(t, p^t) \triangleq \inf_{\tau \geq t+1, \delta} \langle l(\tau, \delta; \theta, D) \mid D > t, p^t \rangle_{\theta, D, \mathbf{x}}. \tag{3}$$

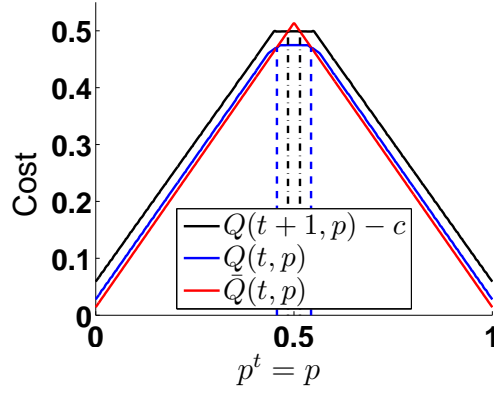

Figure 1: Comparison of the cost $\overline{Q}(t, p)$ of stopping at time $t$ (red); the cost $Q(t, p)$ of continuing at time $t$ (blue solid line); and $Q(t + 1, p) - c$ (black solid line), which is the cost of continuing at time $t + 1$ minus an adjustment $\overline{Q}(t+1, p) - \overline{Q}(t, p) = c$. The continuation region $C^t$ is the interval between the intersections of the solid blue and red lines, marked by the blue dotted lines, and the continuation region $C^{t+1}$ is the interval between the intersections of the solid black and red lines, marked by the black dotted lines. Note that $Q(t + 1, p) - c \geq Q(t, p)$, so $C^t$ contains $C^{t+1}$.

Note that, in general, both $V(t, p^t)$ and $Q(t, p^t)$ may be difficult to compute due to the need to optimize over infinitely many decision policies. Conversely, the cost associated with stopping at time $t$, known as the *Q-factor for stopping* and denoted by $\overline{Q}$, is easily computed as

$$\overline{Q}(t, p^t) = \inf_{\delta=0,1} \langle l(t, \delta; \theta, D) \mid D > t, p^t \rangle_{\theta, D, \mathbf{x}} = \min\{p^t, 1 - p^t\} + ct, \tag{4}$$

where the infimum is obtained by choosing $\delta = 0$ if $p^t \leq .5$ and choosing $\delta = 1$ otherwise.

An optimal stopping rule is to stop the first time the expected cost of continuing exceeds that of stopping, and to choose $\delta = 0$ or $\delta = 1$ to minimize the probability of error given the accumulated evidence (see [10]). That is, $\tau^* = \inf\{t \geq 0 : \overline{Q}(t, p^t) \leq Q(t, p^t)\}$ and $\delta^* = \mathbf{1}_{\{p^{\tau^*} \geq 1/2\}}$. We define the continuation region at time $t$ by $C^t \triangleq \{p^t \in [0, 1] : Q(t, p^t) > \overline{Q}(t, p^t)\}$ so that $\tau^* = \inf\{t \geq 0 : p^t \notin C^t\}$. Although we have obtained an expression for the optimal policy in terms of $Q(t, p)$ and $\overline{Q}(t, p)$, computing $Q(t, p)$ is difficult in general.

**Lemma 1.** *The function $p \mapsto Q(t, p^t)$ is concave with respect to $p^t$ for each $t \in \mathbb{N}$.*
*Proof.* We may restrict the infimum in Eq. 3 to be over only those $\tau$ and $\delta$ depeding on $D$ and the future observations $\mathbf{x}_{t+1} \triangleq \{x^{t+1}, x^{t+2}, \dots\}$. This is due to two facts. First, the expectation is conditioned on $p^t$, which contains all the information about $\theta$ available in the past observations $\mathbf{x}^t$, and makes it unnecessary for the optimal policy to depend on $\mathbf{x}^t$ except through $p^t$. Second, dependence on $p^t$ in the optimal policy may be made implicit by allowing the infimum to be attained by different $\tau$ and $\delta$ for different values of $p^t$ but removing explicit dependence on $p^t$ from the individual policies over which the infimum is taken. With $\tau$ and $\delta$ chosen from this restricted set of policies, we note that the distribution of the future observations $\mathbf{x}_{t+1}$ is entirely determined by $\theta$ and so we have $\langle l(\tau, \delta; \theta, D) \mid \theta, p^t \rangle_{D, \mathbf{x}_{t+1}} = \langle l(\tau, \delta; \theta, D) \mid \theta \rangle_{D, \mathbf{x}_{t+1}}$. Summing over the possible values of $\theta$, we may then write:

$$\langle l(\tau, \delta; \theta, D) \mid p^t \rangle_{\theta, D, \mathbf{x}_{t+1}} = \sum_{k \in \{0,1\}} \langle l(\tau, \delta; \theta, D) \mid \theta = k \rangle_{D, \mathbf{x}_{t+1}} \mathbb{P}\{\theta = k \mid p^t\}$$

$$= \langle l(\tau, \delta; \theta, D) \mid \theta = 0 \rangle_{D, \mathbf{x}_{t+1}} (1 - p^t) + \langle l(\tau, \delta; \theta, D) \mid \theta = 1 \rangle_{D, \mathbf{x}_{t+1}} p^t.$$

Eq. (3) can then be rewritten as:

$$Q(t, p^t) = \inf_{\tau \geq t+1, \delta} \langle l(\tau, \delta; \theta, D) \mid \theta = 0 \rangle_{D, \mathbf{x}_{t+1}} (1 - p^t) + \langle l(\tau, \delta; \theta, D) \mid \theta = 1 \rangle_{D, \mathbf{x}_{t+1}} p^t,$$

where this infimum is again understood to be taken over this set of policies depending only upon observations after time $t$. Since neither $\langle l(\tau, \delta; \theta, D) \mid \theta = 0 \rangle$ nor $\langle l(\tau, \delta; \theta, D) \mid \theta = 1 \rangle$ depend on $p^t$, this is the infimum of a collection of linear functions in $p^t$, and hence is concave in $p^t$ ( [9]). $\square$

We now need a lemma describing how expected cost depends on the distribution of the deadline. Let $D'$ be a deadline whose distribution is different than that of $D$. Let $\pi^*$ be the policy that is optimal given that the deadline has distribution $D$, and denote $\sigma_{\pi^*}$ by $\sigma^*$. Then define

$$V'(t, p^t) \triangleq \langle \min(p^{\sigma^*}, 1 - p^{\sigma^*}) \mathbf{1}_{\{\sigma^* < D'\}} + c \min(\sigma^*, D') + d\mathbf{1}_{\{\sigma^* \geq D'\}} \mid p^t, D' > t \rangle_{\theta, D, \mathbf{x}}$$

so that $V'$ gives the expected cost of taking the stopping time $\sigma^*$ which is optimal for deadline $D$ and applying it to the situation with deadline $D'$. Similarly, let $Q'(t, p^t)$ and $\overline{Q}'(t, p^t)$ denote the corresponding expected costs under $\sigma^*$ and $D'$ given that we continue or stop, respectively, at time $t$ given $p^t$ and $D' > t$. Note that $\overline{Q}'(t, p^t) = \overline{Q}(t, p^t) = \min(p^t, 1 - p^t) + ct$. These definitions are the basis for the following lemma, which essentially shows that replacing the deadline $D$ which a less urgent deadline $D'$ lowers cost. This lemma is needed for Lemma 3 below.

**Lemma 2**  *If $D'$ is such that $\mathbb{P}\{D' > t+1 \mid D' > t\} \geq \mathbb{P}\{D > t+1 \mid D > t\}$ for all $t$, then $V'(t, p) \leq V(t, p)$ and $Q'(t, p) \leq Q(t, p)$ for all $t$ and $p$.*
*Proof.* First let us show that if we have $V'(t+1, p') \leq V(t+1, p')$ for some fixed $t$ and all $p'$, then we also have $Q'(t, p) \leq Q(t, p)$ for that same $t$ and all $p$. This is the case because, if we fix $t$, then

$$
\begin{aligned}
Q(t, p^t) &= (d + c(t+1))\, \mathbb{P}\{D{=}t{+}1 \mid D{>}t\} + \langle V(t+1, p^{t+1}) \mid p^t \rangle_{x^{t+1}} \mathbb{P}\{D{>}t{+}1 \mid D{>}t\} \\
&= d + c(t+1) + \langle V(t+1, p^{t+1}) - (d + c(t+1)) \mid p^t \rangle_{x^{t+1}} \mathbb{P}\{D{>}t{+}1 \mid D{>}t\} \\
&\geq d + c(t+1) + \langle V(t+1, p^{t+1}) - (d + c(t+1)) \mid p^t \rangle_{x^{t+1}} \mathbb{P}\{D'{>}t{+}1 \mid D'{>}t\} \\
&\geq d + c(t+1) + \langle V'(t+1, p^{t+1}) - (d + c(t+1)) \mid p^t \rangle_{x^{t+1}} \mathbb{P}\{D'{>}t{+}1 \mid D'{>}t\} = Q'(t, p).
\end{aligned}
$$

In the first inequality we have used two facts: that $V(t+1, p^{t+1}) \leq \overline{Q}(t+1, p^{t+1}) = \min(p^{t+1}, 1 - p^{t+1}) + c(t+1) \leq d + c(t+1)$ (which is true because $d \geq .5$); and that $\mathbb{P}\{D > t+1 \mid D > t\} \leq \mathbb{P}\{D' > t+1 \mid D' > t\}$. In the second inequality we have used our assumption that $V'(t+1, p') \leq V(t+1, p')$ for all $p'$.

Now consider a finite horizon version of the problem where $\sigma^*$ is only optimal among stopping times bounded above by a finite integer $T$. We will show the lemma for this case, and the lemma for the infinite horizon version of the problem follows by taking the limit as $T \to \infty$.

We induct backwards on $t$. Since $\sigma^*$ is required to stop at $T$, we have $V(T, p^T) = \overline{Q}(T, p^T) = \overline{Q}'(T, p^T) = V'(T, p^T)$. Now for the induction step. Fix $p$ and $t < T$. If $\sigma^*$ chooses to stop at $t$ when $p^t = p$, then $V(t, p) = \overline{Q}(t, p) = \overline{Q}'(t, p) = V'(t, p)$. If $\sigma^*$ continues instead, then $V(t, p) = Q(t, p) \geq Q'(t, p) = V'(t, p)$ by the induction hypothesis. $\square$

Note the requirement that $d \geq 1/2$ in the previous lemma. If this requirement is not met, then if $p^t$ is such that $d < \min(p^t, 1 - p^t)$ then we may prefer to get timed out rather than choose $\delta = 0$ or $\delta = 1$ and suffer the expected penalty of $\min(p^t, 1 - p^t)$ for choosing incorrectly. In this situation, since the conditional probability $\mathbb{P}\{D = t+1 \mid D > t\}$ that we will time out in the next time period grows as time moves forward, the continuation region may expand with time rather than contract. Under most circumstances, however, it seems reasonable to assume the deadline cost to be at least as large as that of making an error.

We now state Lemma 3, which shows that the cost of delaying by one time period is as least as large as the continuation cost $c$, but may be larger because the delay causes the deadline to approach more rapidly.

**Lemma 3.**  *For each $t \in \mathbb{N}$ and $p \in (0, 1)$, $Q(t-1, p^{t-1} {=} p) \leq Q(t, p^t {=} p) - c$.*
*Proof.* Fix $t$. Let $\sigma^* \triangleq \inf\{s \geq t+1 : p^s \notin C^s\}$ so that $\min(\sigma^*, D)$ attains the infimum for $Q(t, p^t)$. Also define $\sigma' \triangleq \inf\{s \geq t : p^s \notin C^{s+1}\}$ and $\tau' \triangleq \min(D, \sigma')$. Since $\tau'$ is within the set over which the infimum defining $Q(t-1, p)$ is taken,

$$
\begin{aligned}
Q(t-1, p) &\leq \langle \min(p^{\tau'}, 1 - p^{\tau'}) \mathbf{1}_{\{\tau' < D\}} + c\tau' + d\mathbf{1}_{\{\tau' \geq D\}} \mid D > t-1, p^{t-1} = p \rangle_{D, \mathbf{x}_t} \\
&= \langle \min(p^{\sigma'}, 1 - p^{\sigma'}) \mathbf{1}_{\{\sigma' < D\}} + c\min(D, \sigma') + d\mathbf{1}_{\{\sigma' \geq D\}} \mid D > t-1, p^{t-1} = p \rangle_{D, \mathbf{x}_t} \\
&= \langle \min(p^{\sigma^*}, 1 - p^{\sigma^*}) \mathbf{1}_{\{\sigma^* - 1 < D\}} + c\min(D, \sigma^* - 1) + d\mathbf{1}_{\{\sigma^* - 1 \geq D\}} \mid D > t-1, p^t = p \rangle_{D, \mathbf{x}_{t+1}},
\end{aligned}
$$

where the last step is justified by the stationarity of the observation process, which implies that the joint distribution of $(p^s)_{s \geq t}$, $p^{\sigma^*}$, and $\sigma^*$ conditioned on $p^t = p$ is the same as the joint distribution

of $(p^{s-1})_{s\geq t}$, $p^{\sigma'}$, and $\sigma'+1$ conditioned on $p^{t-1}=p$. Let $D'=D+1$ and we have

$$Q'(t,p) = \langle \min(p^{\sigma^*}, 1-p^{\sigma^*})\mathbf{1}_{\{\sigma^*<D'\}} + c\min(D',\sigma^*) + d\mathbf{1}_{\{\sigma^*\geq D'\}} \mid D' > t, p^t = p\rangle_{D',\mathbf{x}_{t+1}},$$

so $Q(t-1,p) \leq Q'(t,p) - c$. Finally, as $D'$ satisfies the requirements of Lemma 2, $Q'(t,p) \leq Q(t,p)$. $\square$

**Lemma 4.** *For $t\in\mathbb{N}$, $Q(t,0) = Q(t,1) = c(t+1) + d\mathbb{P}\{D=t+1\mid D>t\}$.*
*Proof.* On the event $p^t=0$, we have that $\mathbb{P}\{\theta=0\}=1$ and the policy attaining the infimum in (3) is $\tau^*=t+1$, $\delta^*=0$. Thus, $Q(t,0)$ becomes

$$Q(t,0) = \langle l(\tau^*,\delta^*;\theta,D) \mid D>t, p^t=0\rangle_{D,\mathbf{x}_{t+1}} = \langle l(\tau^*,\delta^*;\theta,D) \mid D>t,\theta=0\rangle_{D,\mathbf{x}_{t+1}}$$
$$= \langle d\mathbf{1}_{\{t+1\geq D\}} + c(t+1) \mid D>t,\theta=0\rangle_{D,\mathbf{x}_{t+1}} = c(t+1) + d\mathbb{P}\{D=t+1\mid D>t\}.$$

Similarly, on the event $p^t=1$, we have that $\mathbb{P}\{\theta=1\}=1$ and the policy attaining the infimum in (3) is $\tau^*=t+1$, $\delta^*=1$. Thus, $Q(t,1) = c(t+1) + d\mathbb{P}\{D\leq t+1\mid D>t\}$. $\square$

We are now ready for the main theorem, which shows that $C^t$ is either empty or an interval, and that $C^{t+1}\subseteq C^t$. To illustrate our proof technique, we plot $Q(t,p)$, $\overline{Q}(t,p)$, and $Q(t+1,p)-c$ as functions of $p$ in Figure 2.1. As noted, the continuation region $C^t$ is the set of $p$ such that $Q(t,p)\leq \overline{Q}(t,p)$, To show that $C^t$ is either empty or an interval, we note that $Q(t,p)$ is a concave function in $p$ (Lemma 1) whose value at the endpoints $p=0,1$ are greater than the corresponding values of $\overline{Q}(t,p)$ (Lemma 4). Such a concave function may only intersect $\overline{Q}(t,p)$, which is a constant plus $\min(p,1-p)$, either twice or not at all. When it intersects twice, we have the situation pictured in Figure 2.1, in which $C^t$ is a non-empty interval, and when it does not intersect $C^t$ is empty.

To show that $C^{t+1}\subseteq C^t$ we note that the difference between $\overline{Q}(t+1,p)$ and $\overline{Q}(t,p)$ is the constant $c$. Thus, to show that $C^t$, the set where $\overline{Q}(t,p)$ contains $Q(t,p)$, is larger than $C^{t+1}$, the set where $\overline{Q}(t+1,p)$ is larger than $Q(t+1,p)$, it is enough to show that the difference between $Q(t+1,p)$ and $Q(t,p)$ is at least as large as the adjustment $c$, which we have done in Lemma 3.

**Theorem.** *At each time $t\in\mathbb{N}$, the optimal continuation region $C^t$ is either empty or a closed interval, and $C^{t+1}\subseteq C^t$.*
*Proof.* Fix $t\in\mathbb{N}$. We begin by showing that $C^{t+1}\subseteq C^t$. If $C^{t+1}$ is empty then the statement follows trivially, so consider the case when $C^{t+1}\neq\emptyset$. Choose $p\in C^{t+1}$. Then

$$Q(t,p) \leq Q(t+1,p) - c \leq \overline{Q}(t+1,p) - c = \min\{p,1-p\} + ct = \overline{Q}(t,p).$$

Thus, $p\in C^t$, implying $C^{t+1}\subseteq C^t$.

Now suppose that $C^t$ is non-empty and we will show it must be a closed interval. Let $a^t \triangleq \inf C^t$ and $b^t \triangleq \sup C^t$. Since $C^t$ is a non-empty subset of $[0,1]$, we have $a^t, b^t \in [0,1]$. Furthermore, $a^t > 0$ because $Q(t,p) \geq c(t+1) + d\mathbb{P}\{D=t+1\mid D>t\} > ct = \overline{Q}(t,0)$ for all p, and the continuity of $\overline{Q}(t,\cdot)$ implies that $Q(t,p) > \overline{Q}(t,p) > 0$ for $p$ in some open interval around 0. Similarly, $b^t < 1$. Thus, $a^t, b^t \in (0,1)$.

We will show first that $[a^t, 1/2]\subseteq C^t$. If $a^t > 1/2$ then this is trivially true, so consider the case that $a^t \leq 1/2$. Since $Q(t,\cdot)$ is concave on the open interval $(0,1)$, it must also be continuous there. This and the continuity of $\overline{Q}$ imply that $Q(t,a^t) = \overline{Q}(t,a^t)$. Also, $Q(t,0) > \overline{Q}(t,0)$ by Lemma 4. Thus $a^t > 0$ and we may take a left-derivative at $a^t$. For any $\varepsilon \in (0,a^t)$, $a^t - \varepsilon \notin C^t$ so $Q(a^t-\varepsilon) > \overline{Q}(a^t-\varepsilon)$. This implies together with $Q(t,a^t) = \overline{Q}(t,a^t)$ that

$$\frac{\partial^-}{\partial p}Q(t,a^t) = \lim_{\varepsilon\to 0^+} \frac{Q(t,a^t) - Q(t,a^t-\varepsilon)}{\varepsilon} \leq \lim_{\varepsilon\to 0^+} \frac{\overline{Q}(t,a^t) - \overline{Q}(t,a^t-\varepsilon)}{\varepsilon} = \frac{\partial^-}{\partial p}\overline{Q}(t,a^t).$$

Since $Q(t,\cdot)$ is concave by Lemma 1 and $\overline{Q}(t,\cdot)$ is linear on $[0,1/2]$, we have for any $p'\in[a^t,1/2]$,

$$\frac{\partial^-}{\partial p}Q(t,p') \leq \frac{\partial^-}{\partial p}Q(t,a^t) \leq \frac{\partial^-}{\partial p}\overline{Q}(t,a^t) = \frac{\partial^-}{\partial p}\overline{Q}(t,p').$$

Since $Q(t,\cdot)$ is concave, it is differentiable except at countably many points, so for any $p\in[a^t,1/2]$,

$$Q(t,p) = Q(t,a^t) + \int_{a^t}^{p} \frac{\partial^-}{\partial p}Q(t,p')\,dp' \leq \overline{Q}(t,a^t) + \int_{a^t}^{p} \frac{\partial^-}{\partial p}\overline{Q}(t,p')\,dp' = \overline{Q}(t,p).$$

Therefore $p \in C^t$, and, more generally, $[a^t, 1/2] \subseteq C^t$. By a similar argument, $[1/2, b^t] \subseteq C^t$.

Finally, $C^t \subseteq [a^t, b^t] \subseteq [a^t, 1/2] \cup [1/2, b^t] \subseteq C^t$ and we must have $C^t = [a^t, b^t]$. $\square$

We also include the following proposition, which shows that if $D$ is finite with probability 1 then the continuation region must eventually narrow to nothing.

**Proposition.** *If* $\mathbb{P}\{D < \infty\} = 1$ *then there exists a* $T < \infty$ *such that* $C^T = \emptyset$.
*Proof.* First consider the case when $D$ is bounded, so $\mathbb{P}\{D \leq T+1\} = 1$ for some time $T < \infty$. Then, $Q(T, p^T) = d + c(T+1)$, while $\overline{Q}(T, p^T) = cT + \min(p^T, 1 - p^T) \leq cT + 1/2$. Thus $Q(T, p^T) - \overline{Q}(T, p^T) \geq d + c - 1/2 > 0$, and $C^T = \emptyset$.

Now consider the case when $\mathbb{P}\{D > t\} > 0$ for every $t$. By neglecting the error probability and including only continuation and deadline costs, we obtain $Q(t, p^t) \geq d\,\mathbb{P}\{D = t+1 \mid D > t\} + c(t+1)$. Bounding the error probability by $1/2$ we obtain $\overline{Q}(t, p^t) \leq ct + 1/2$. Thus, $Q(t, p^t) - \overline{Q}(t, p^t) \geq c + d\,\mathbb{P}\{D = t+1 \mid D > t\} - 1/2$. Since $\mathbb{P}\{D < \infty\} = 1$, $\lim_{t \to \infty} c + d\,\mathbb{P}\{D = t+1 \mid D > t\} - 1/2 = c + d - 1/2 > 0$, and there exists a $T$ such that $c + d\,\mathbb{P}\{D = t+1 \mid D > t\} - 1/2 > 0$ for every $t \geq T$. This implies that, for $t \geq T$ and $p^t \in [0, 1]$, $Q(t, p^t) - \overline{Q}(t, p^t) > 0$ and $C^t = \emptyset$. $\square$

## 3  Computational simulations

We conducted a series of simulations in which we computed the continuation region and distributions of response time and accuracy for the optimal policy for several choices of the parameters $c$ and $d$, and for the distribution of the deadline $D$. We chose the observation $x_t$ to be a Bernoulli random variable under both $f_0$ and $f_1$ for every $t = 1, 2, \ldots$ with different values for $q_\theta \triangleq \mathbb{P}\{x_i = 1 \mid \theta\}$. In our simulations we chose $q_0 = .45$ and $q_1 = .55$.

We computed optimal policies for two different forms of deadline distribution: first for a deterministic deadline fixed to some known constant; and second for a gamma distributed deadline. The gamma distribution with parameters $k > 0$ and $\beta > 0$ has density $(\beta^k / \Gamma(k)) x^{k-1} e^{-\beta x}$ for $x > 0$, where $\Gamma(\cdot)$ is the gamma function. The parameters $k$ and $\beta$, called the *shape* and *rate* parameters respectively, are completely determined by choosing the mean and the standard deviation of the distribution since the gamma distribution has mean $k/\beta$ and variance $k/\beta^2$. A fixed deadline $T$ may actually be seen as a limiting case of a gamma-distributed deadline by taking both $k$ and $\beta$ to infinity such that $k/\beta = T$ is fixed.

We used the table-look-up form of the backward dynamic programming algorithm (see, e.g., [11]) to compute the optimal Q-factors. We obtained approximations of the value function and Q-factors at a finite set of equally spaced discrete points $\{0, 1/N, \ldots, (N-1)/N, 1\}$ in the interval $[0, 1]$. In our simulations we chose $N = 999$. We establish a final time $T$ that is large enough that $\mathbb{P}\{D \leq T\}$ is nearly 1, and thus $\mathbb{P}\{\tau^* \leq T\}$ is also nearly 1. In our simulations we chose $T = 60$. We approximated the value function $V(T, p^T)$ at this final time by $\overline{Q}(T, p^T)$. Then we calculated value functions and Q-factors for previous times recursively according to Bellman's equation:

$$Q(t, p) = \langle V(t+1, p^{t+1}) \mid p^t = p \rangle_{p^{t+1}}; \qquad V(t, p) = \min(\overline{Q}(t, p), Q(t, p)).$$

This expectation relating $Q(t, \cdot)$ to $V(t+1, \cdot)$ may be written explicitly using our hypotheses and Eq. 1 to define a function $g$ so that $p^{t+1} = g(p^t, x^{t+1})$. In our case this function is defined by $g(p^t, 1) \triangleq (p^t q_1)/(p^t q_1 + (1 - p^t) q_0)$ and $g(p^t, 0) \triangleq (p^t (1 - q_1))/(p^t (1 - q_1) + (1 - p^t)(1 - q_0))$. Then we note that $\mathbb{P}\{x^{t+1} = 1 \mid p^t\} = \mathbb{P}\{x^{t+1} = 1 \mid \theta = 1\} p^t + \mathbb{P}\{x^{t+1} = 1 \mid \theta = 0\}(1 - p^t) = p^t q_1 + (1 - p^t) q_0$, and similarly $\mathbb{P}\{x^{t+1} = 0 \mid p^t\} = p^t(1 - q_1) + (1 - p^t)(1 - q_0)$. Then

$$Q(t, p^t) = (c(t+1) + d)\mathbb{P}\{D \leq t+1 \mid D > t\} + \mathbb{P}\{D > t+1 \mid D > t\}\big[$$
$$V\big(t+1, g(p^t, 1)\big)\big(p^t q_1 + (1 - p^t) q_0\big) + V\big(t+1, g(p^t, 0)\big)\big(p^t(1 - q_1) + (1 - p^t)(1 - q_0)\big)\big].$$

We computed continuation regions $C^t$ from these Q-factors, and then used Monte Carlo simulation with $10^6$ samples for each problem setting to estimate $\mathbb{P}\{\delta = \theta \mid \tau = t\}$ and $\mathbb{P}\{\tau = t\}$ as functions of $t$. The results of these computational simulations are shown in Figure 3. We see in Fig. 3A that the decision boundaries for a fixed deadline (solid blue) are smoothly narrowing toward the midline. Clearly, at the last opportunity for responding before the deadline, the optimal policy would always generate a response (and therefore the thresholds merge), since we assumed that the cost of penalty

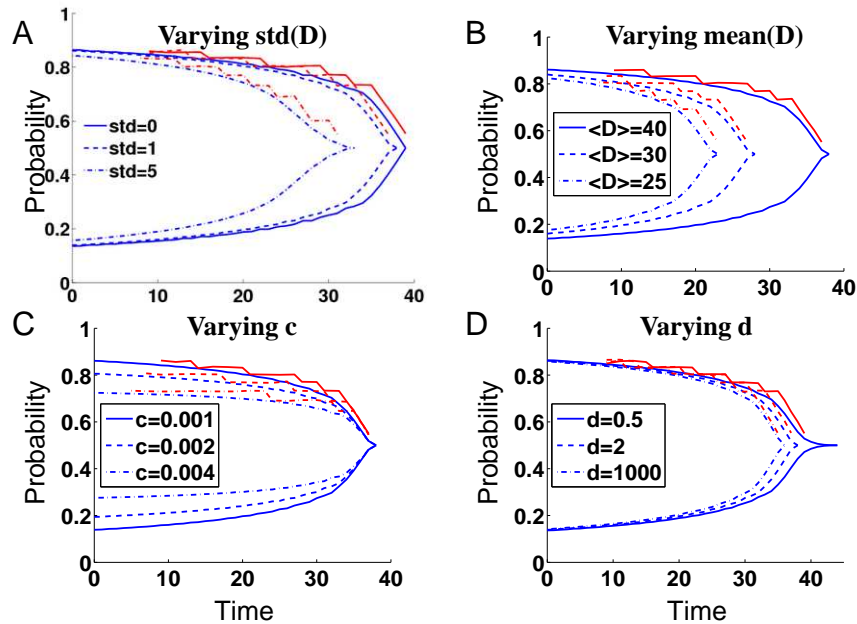

Figure 2: Plots of the continuation region $C^t$ (blue), and the probability of a correct response $\mathbb{P}\{\delta = \theta \mid \tau = t\}$ (red). The default settings were $c = .001$, $d = 2$, mean$(D) = 40$, std$(D) = 1$, and $q_0 = 1 - q_1 = .45$. In each plot we varied one of them while keeping the others fixed. In (A) we varied the standard deviation of D, in (B) the mean of D, in (C) the value of $c$, and in (D) the value of $d$.

is greater than the expected cost of making an error: $d \geq .5$ (since the optimal policy is to choose the hypothesis with probability $\geq .5$, the expected probability of error is always $\leq .5$). At the time step before, the optimal policy would only continue if one more data point is going to improve the belief state enough to offset the extra time cost $c$. Therefore, the optimal policy only continues for a small "window" around .5 even though it has the opportunity to observe one more data point. At earlier times, the window "widens" following similar logic. When uncertainty about the deadline increases (larger std$(D)$; shown in dashed and dash-dotted blue lines), the optimal thresholds are squeezed toward each other and to the left, the intuition being that the threat of encountering the deadline spreads earlier and earlier into the trial. The red lines denote the average accuracy for different stopping times obtained from a million Monte Carlo simulations of the observation-decision process. They closely follow the decision thresholds (since the threshold is on the posterior probability $p^\tau$), but are slightly larger, because $p^\tau$ must exceed the threshold, and $p^t$ moves in discrete increments due to the discrete Bernoulli process.

The effect of decreasing the mean deadline is to shift the decision boundaries left-ward, as shown in Fig. 3B. The effect of increasing the cost of time $c$ is to squeeze the boundaries toward the midline (Fig. 3C – this result is analogous to that seen in the classical sequential probability ratio test for the infinite-horizon case. The effect of increasing $d$ is to squeeze the thresholds to the left (Fig. 3D), and the rate of shifting is on the order of $\log(d)$ because the tail of the gamma distribution is falling off nearly exponentially.

## 4  Discussion

In this work, we formalized the problem of sequential hypothesis testing (of two alternatives) under the pressure of a stochastically sampled deadline, and characterized the optimal policy. For a large class of deadline distributions (including gamma, normal, exponential, delta), we showed that the optimal policy is to report a hypothesis as soon as the posterior belief hits one of a pair of monotonically declining thresholds (toward the midline). This generalizes the classical infinite horizon case in the limit when the deadline goes to infinity, and the optimal policy reverts to a pair of fixed thresholds as in the sequential probability ratio test [1]. We showed that the decision policy becomes more conservative (thresholds pushed outward and to the right) when there's less uncertainty about

the deadline, when the mean of the deadline is larger, when the linear temporal cost is larger, and when the deadline cost is smaller.

In the theoretical analysis, we assumed that $D$ has the property that $\mathbb{P}\{D > t+u \mid D > t\}$ is non-increasing in $t$ for each $u \geq 0$ over the set of $t$ such that $\mathbb{P}\{D > t\} > 0$. This assumption implies that, if the deadline has not occurred already, then the likelihood that it will happen soon grows larger and larger, as time passes. The assumption is violated by multi-modal distributions, for which there is a large probability the deadline will occur at some early point in time, but if the deadline does not occur by that point in time then will not occur until some much later time. This assumption is met by a fixed deadline (std$(D) \to 0$), and also includes the classical infinite-horizon case ($D \to \infty$) as a special case (and the optimal policy reverts to the sequential probability ratio test). This assumption is also met by any distribution with a log-concave density because $\log \mathbb{P}\{D > t+u \mid D > t\} = \log \mathbb{P}\{D > t+u\} - \log \mathbb{P}\{D > t\} = F(t+u) - F(t)$, where $F(t) \triangleq \log \mathbb{P}\{D > t\}$. If the density of $D$ is log-concave, then $F$ is concave ( [9]), and the increment $F(t+u) - F(t)$ is non-increasing in $t$. Many common distributions have log-concave densities, including the exponential distribution, the gamma distribution, the normal distribution, and the uniform distribution on an interval.

We used gamma distributions for the deadline in the numerical stimulations. There are several empirical properties about timing uncertainty in humans and animals that make the gamma distribution particularly suitable. First, realizations from the gamma distribution are always non-negative, which is consistent with the assumption that a subject never thinks a deadline has passed before the experiment has started. Second, if we fix the rate parameter $\beta$ and vary the shape $k$, then we obtain a collection of deadline distributions with different means whose variance and mean are in a fixed ratio, which is consistent with experimental observations [12]. Third, for large values of $k$ the gamma distribution is approximately normal, which is also consistent with experimental observations [12]. Finally, a gamma distributed random variable with mean $\mu$ may be written as the sum of $k = \mu\beta$ independent exponential random variables with mean $1/\beta$, so if the brain were able to construct an exponential-distributed timer whose mean $1/\beta$ were on the order of milliseconds, then it could construct a very accurate gamma-distributed timer for intervals of several seconds by resetting this exponential timer $k$ times and responding after the $k$th alarm. This has interesting ramifications for how sophisticated timers for relatively long intervals can be constructed from neurons that exhibit dynamics on the order of milliseconds.

This work makes several interesting empirical predictions. Subjects who have more internal uncertainty, and therefore larger variance in their perceived deadline stochasticity, should respond to stimuli earlier and with lower accuracy. Similarly, the model makes quantitative predictions about the subject's performance when the experimenter explicitly manipulates the mean deadline, and the relative costs of error, time, and deadline.

## Acknowledgments

We thank Jonathan Cohen, Savas Dayanik, Philip Holmes, and Warren Powell for helpful discussions. The first author was supported in part by the Air Force Office of Scientific Research under grant AFOSR-FA9550-05-1-0121.

## References

[1] Wald, A & Wolfowitz, J (1948). *Ann. Math. Statisti.* **19**: 326-39.

[2] Luce, R D (1986). *Response Times: Their Role in Inferring Elementary Mental Org.* Oxford Univ. Press.

[3] Ratcliff, R & Rouder, J N (1998). *Psychol. Sci.* **9**: 347-56.

[4] Bogacz, R *et al* (2006). *Pyschol. Rev.* **113**: 700-65.

[5] Bertsekas, D P (1995). *Dynamic Programming and Optimal Control.* Athena Scientific.

[6] Gold, J I & Shadlen, M N (2002). *Neuron* **36**: 299-308.

[7] Mozer et al (2004). *Proc. Twenty Sixth Annual Conference of the Cognitive Science Society.* 981-86.

[8] Siegmund, D (1985). *Sequential Analysis.* Springer.

[9] Boyd, S & Vandenberghe, L (2004) *Convex Optimization.* Cambridge Univ. Press.

[10] Poor, H V (1994). *An Introduction to Signal Detection and Estimation.* Springer-Verlag.

[11] Powell, W B (2007) *Approximate Dynamic Programming: Solving the curses of dimensionality.* Wiley.

[12] Rakitin, et al (1998). *J. Exp. Psychol. Anim. Behav. Process.* **24**: 15-33.

